# Error-correcting Codes on a Bethe-like Lattice

**Renato Vicente**      **David Saad**
The Neural Computing Research Group
Aston University, Birmingham, B4 7ET, United Kingdom
*{vicenter,saadd}@aston.ac.uk*

**Yoshiyuki Kabashima**
Department of Computational Intelligence and Systems Science
Tokyo Institute of Technology, Yokohama 2268502, Japan
*kaba@dis.titech.ac.jp*

## Abstract

We analyze Gallager codes by employing a simple mean-field approximation that distorts the model geometry and preserves important interactions between sites. The method naturally recovers the probability propagation decoding algorithm as an extremization of a proper free-energy. We find a thermodynamic phase transition that coincides with information theoretical upper-bounds and explain the practical code performance in terms of the free-energy landscape.

## 1   Introduction

In the last years increasing interest has been devoted to the application of mean-field techniques to inference problems. There are many different ways of building mean-field theories. One can make a perturbative expansion around a tractable model [1,2], or assume a tractable structure and variationally determine the model parameters [3].

Error-correcting codes (ECC) are particularly interesting examples of inference problems in loopy intractable graphs [4]. Recently the focus has been directed to the state-of-the art high performance turbo codes [5] and to Gallager and MN codes [6,7]. Statistical physics has been applied to the analysis of ECCs as an alternative to information theory methods yielding some new interesting directions and suggesting new high-performance codes [8]. Sourlas was the first to relate error-correcting codes to spin glass models [9], showing that the Random-energy Model [10] can be thought of as an ideal code capable of saturating Shannon's bound at vanishing code rates. This work was extended recently to the case of finite code rates [11] and has been further developed for analyzing MN codes of various structures [12]. All of the analyzes mentioned above as well as the recent turbo codes analysis [13] relied on the replica approach under the assumption of replica symmetry. To date, the only model that can be analyzed exactly is the REM that corresponds to an impractical coding scheme of a vanishing code rate.

Here we present a statistical physics treatment of non-structured Gallager codes by employing a mean-field approximation based on the use of a generalized tree structure (Bethe

lattice [14]) known as Husimi cactus that is exactly solvable. The model parameters are just assumed to be those of the model with cycles. In this framework the probability propagation decoding algorithm (PP) emerges naturally providing an alternative view to the relationship between PP decoding and mean-field approximations already observed in [15]. Moreover, this approach has the advantage of being a slightly more controlled and easier to understand than replica calculations.

This paper is organized as follows: in the next section we present unstructured Gallager codes and the statistical physics framework to analyze them, in section 3 we make use of the lattice geometry to solve the model exactly. In section 4 we analyze the typical code performance. We summarize the results in section 5.

## 2   Gallager codes: statistical physics formulation

We will concentrate here on a simple communication model whereby messages are represented by binary vectors and are communicated through a Binary Symmetric Channel (BSC) where uncorrelated bit flips appear with probability $f$. A Gallager code is defined by a binary matrix $A = [C_1 \mid C_2]$, concatenating two very sparse matrices known to both sender and receiver, with $C_2$ (of dimensionality $(M - N) \times (M - N)$) being invertible - the matrix $C_1$ is of dimensionality $(M - N) \times N$.

Encoding refers to the production of an $M$ dimensional binary code word $t \in \{0, 1\}^M$ ($M > N$) from the original message $\xi \in \{0, 1\}^N$ by $t = G^T \xi$ (mod 2), where all operations are performed in the field $\{0, 1\}$ and are indicated by (mod 2). The generator matrix is $G = [I \mid C_2^{-1} C_1]$ (mod 2), where $I$ is the $N \times N$ identity matrix, implying that $A G^T$ (mod 2) $= 0$ and that the first $N$ bits of $t$ are set to the message $\xi$. In *regular* Gallager codes the number of non-zero elements in each row of $A$ is chosen to be exactly $K$. The number of elements per column is then $C = (1 - R)K$, where the code rate is $R = N/M$ (for unbiased messages). The encoded vector $t$ is then corrupted by noise represented by the vector $\zeta \in \{0, 1\}^M$ with components independently drawn from $P(\zeta) = (1-f)\delta(\zeta) + f\delta(\zeta - 1)$. The received vector takes the form $r = G^T \xi + \zeta$ (mod 2).

Decoding is carried out by multiplying the received message by the matrix $A$ to produce the *syndrome* vector $z = Ar = A\zeta$ (mod 2) from which an estimate $\hat{\tau}$ for the noise vector can be produced. An estimate for the original message is then obtained as the first $N$ bits of $r + \hat{\tau}$ (mod 2). The Bayes optimal estimator (also known as *marginal posterior maximizer*, MPM) for the noise is defined as $\hat{\tau}_j = \text{argmax}_{\tau_j} P(\tau_j \mid z)$, where $\tau_j \in \{0, 1\}$. The performance of this estimator can be measured by the probability of bit error $p_b = 1 - 1/M \sum_{j=1}^{M} \delta[\hat{\tau}_j; \zeta_j]$, where $\delta[;]$ is Kronecker's delta. Knowing the matrices $C_2$ and $C_1$, the syndrome vector $z$ and the noise level $f$ it is possible to apply Bayes' theorem and compute the posterior probability

$$P(\tau \mid z) = \frac{1}{Z} \chi[z = A\tau (\text{mod } 2)] P(\tau), \qquad (1)$$

where $\chi[X]$ is an indicator function providing 1 if $X$ is true and 0 otherwise. To compute the MPM one has to compute the marginal posterior $P(\tau_j \mid z) = \sum_{i \neq j} P(\tau \mid z)$, which in general requires $\mathcal{O}(2^M)$ operations, thus becoming impractical for long messages. To solve this problem one can use the sparseness of $A$ to design algorithms that require $\mathcal{O}(M)$ operations to perform the same task. One of these methods is the probability propagation algorithm (PP), also known as belief propagation or sum-product algorithm [16].

The connection to statistical physics becomes clear when the field $\{0, 1\}$ is replaced by Ising spins $\{\pm 1\}$ and mod 2 sums by products [9]. The syndrome vector acquires the form of a multi-spin coupling $\mathcal{J}_\mu = \prod_{j \in \mathcal{L}(\mu)} \zeta_j$ where $j = 1, \cdots, M$ and $\mu = 1, \cdots, (M - N)$.

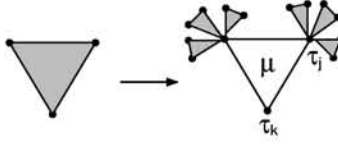

Figure 1: Husimi cactus with $K = 3$ and connectivity $C = 4$.

The $K$ indices of nonzero elements in the row $\mu$ of a matrix $\boldsymbol{A}$, which is not necessarily a concatenation of two separate matrices (therefore, defining an *unstructured* Gallager code), are given by $\mathcal{L}(\mu) = \{j_1, \cdots, j_K\}$, and in a column $l$ are given by $\mathcal{M}(l) = \{\mu_1, \cdots, \mu_C\}$.

The posterior (1) can be written as the Gibbs distribution [12]:

$$P_\beta(\boldsymbol{\tau} \mid \mathcal{J}) = \frac{1}{Z} \lim_{\gamma \to \infty} \exp\left[-\beta \mathcal{H}_\gamma(\boldsymbol{\tau}; \mathcal{J})\right] \tag{2}$$

$$\mathcal{H}_\gamma(\boldsymbol{\tau}; \mathcal{J}) = -\gamma \sum_{\mu=1}^{M-N} \left( \mathcal{J}_\mu \prod_{j \in \mathcal{L}(\mu)} \tau_j - 1 \right) - F \sum_{j=1}^{M} \tau_j \,.$$

The external field corresponds to the prior probability over the noise and has the form $F = \operatorname{atanh}(1 - 2f)$. Note that the Hamiltonian depends on a hyper-parameter that has to be set as $\gamma \to \infty$ for optimal decoding. The disorder is trivial and can be gauged as $\mathcal{J}_\mu \mapsto 1$ by using $\tau_j \mapsto \tau_j \zeta_j$. The resulting Hamiltonian is a multi-spin ferromagnet with finite connectivity in a random field $h_j = F\zeta_j$. The decoding process corresponds to finding local magnetizations at temperature $\beta = 1$, $m_j = \langle \tau_j \rangle_{\beta=1}$ and calculating estimates as $\widehat{\tau}_j = \operatorname{sgn}(m_j)$.

In the $\{\pm 1\}$ representation the probability of bit error, acquires the form

$$p_b = \frac{1}{2} - \frac{1}{2M} \sum_{j=1}^{M} \zeta_j \operatorname{sgn}(m_j), \tag{3}$$

connecting the code performance with the computation of local magnetizations.

## 3 Bethe-like Lattice calculation

### 3.1 Generalized Bethe lattice: the Husimi cactus

A Husimi cactus with connectivity $C$ is generated starting with a polygon of $K$ vertices with one Ising spin in each vertex (generation 0). All spins in a polygon interact through a single coupling $\mathcal{J}_\mu$ and one of them is called the base spin. In figure 1 we show the first step in the construction of a Husimi cactus, in a generic step the base spins of the $n-1$ generation polygons, numbering $(C-1)(K-1)$, are attached to $K-1$ vertices of a generation $n$ polygon. This process is iterated until a maximum generation $n_{\max}$ is reached, the graph is then completed by attaching $C$ uncorrelated branches of $n_{\max}$ generations at their base spins. In that way each spin inside the graph is connected to exactly $C$ polygons. The local magnetization at the centre $m_j$ can be obtained by fixing boundary (initial) conditions in the 0-th generation and iterating recursion equations until generation $n_{\max}$ is reached. Carrying out the calculation in the thermodynamic limit corresponds to having $n_{\max} \sim \ln M$ generations and $M \to \infty$.

The Hamiltonian of the model has the form (2) where $\mathcal{L}(\mu)$ denotes the polygon $\mu$ of the lattice. Due to the tree-like structure, local quantities far from the boundary can be cal-

culated recursively by specifying boundary conditions. The typical decoding performance can therefore be computed exactly without resorting to replica calculations [17].

## 3.2 Recursion relations: probability propagation

We adopt the approach presented in [18] where recursion relations for the probability distribution $P_{\mu k}(\tau_k)$ of the base spin of the polygon $\mu$ is connected to $(C-1)(K-1)$ distributions $P_{\nu j}(\tau_j)$, with $\nu \in \mathcal{M}(j) \setminus \mu$ (all polygons linked to $j$ but $\mu$) of polygons in the previous generation:

$$P_{\mu k}(\tau_k) = \frac{1}{\mathcal{N}} \operatorname{Tr}_{\{\tau_j\}} \exp \left[ \beta \left( \mathcal{J}_\mu \tau_k \prod_{j \in \mathcal{L}(\mu) \setminus k} \tau_j - 1 \right) + F\tau_k \right] \prod_{\nu \in \mathcal{M}(j) \setminus \mu} \prod_{j \in \mathcal{L}(\mu) \setminus k} P_{\nu j}(\tau_j),$$
(4)

where the trace is over the spins $\tau_j$ such that $j \in \mathcal{L}(\mu) \setminus k$.

The effective field $\widehat{x}_{\nu j}$ on a base spin $j$ due to neighbors in polygon $\nu$ can be written as :

$$\exp\left(-2\widehat{x}_{\nu j}\right) = e^{2F} \frac{P_{\nu j}(-)}{P_{\nu j}(+)},$$
(5)

Combining (4) and (5) one finds the recursion relation:

$$\exp\left(-2\widehat{x}_{\mu k}\right) = \frac{\operatorname{Tr}_{\{\tau_j\}} \exp\left[-\beta \mathcal{J}_\mu \prod_{j \in \mathcal{L}(\mu) \setminus k} \tau_j + \sum_{j \in \mathcal{L}(\mu) \setminus k} (F + \sum_{\nu \in \mathcal{M}(j) \setminus \mu} \widehat{x}_{\nu j})\tau_j\right]}{\operatorname{Tr}_{\{\tau_j\}} \exp\left[+\beta \mathcal{J}_\mu \prod_{j \in \mathcal{L}(\mu) \setminus k} \tau_j + \sum_{j \in \mathcal{L}(\mu) \setminus k} (F + \sum_{\nu \in \mathcal{M}(j) \setminus \mu} \widehat{x}_{\nu j})\tau_j\right]}$$
(6)

By computing the traces and taking $\beta \to \infty$ one obtains:

$$\widehat{x}_{\mu k} = \operatorname{atanh} \left[ \mathcal{J}_\mu \prod_{j \in \mathcal{L}(\mu) \setminus k} \tanh(F + \sum_{\nu \in \mathcal{M}(j) \setminus \mu} \widehat{x}_{\nu j}) \right]$$
(7)

The effective local magnetization due to interactions with the nearest neighbors in one branch is given by $\widehat{m}_{\mu j} = \tanh(\widehat{x}_{\mu j})$. The effective local field on a base spin $j$ of a polygon $\mu$ due to $C-1$ branches in the previous generation and due to the external field is $x_{\mu j} = F + \sum_{\nu \in \mathcal{M}(j) \setminus \mu} \widehat{x}_{\nu j}$; the effective local magnetization is, therefore, $m_{\mu j} = \tanh(x_{\mu j})$. Equation (7) can then be rewritten in terms of $\widehat{m}_{\mu j}$ and $m_{\mu j}$ and the PP equations [7,15,16] can be recovered:

$$m_{\mu k} = \tanh \left( F + \sum_{\nu \in \mathcal{M}(j) \setminus \mu} \operatorname{atanh}\left(\widehat{m}_{\nu k}\right) \right) \qquad \widehat{m}_{\mu k} = \mathcal{J}_\mu \prod_{j \in \mathcal{L}(\mu) \setminus k} m_{\mu j} \qquad (8)$$

Once the magnetizations on the boundary (0-th generation) are assigned, the local magnetization $m_j$ in the central site is determined by iterating (8) and computing :

$$m_j = \tanh \left( F + \sum_{\nu \in \mathcal{M}(j)} \operatorname{atanh}\left(\widehat{m}_{\nu j}\right) \right)$$
(9)

## 3.3 Probability propagation as extremization of a free-energy

The equations (8) describing PP decoding represent extrema of the following free-energy:

$$\mathcal{F}(\{m_{\mu k}, \widehat{m}_{\mu k}\}) = \sum_{\mu=1}^{M-N} \sum_{i \in \mathcal{L}} \ln(1 + m_{\mu i}\widehat{m}_{\mu i}) - \sum_{\mu=1}^{M-N} \ln(1 + \mathcal{J}_\mu \prod_{i \in \mathcal{L}} m_{\mu i}) \quad (10)$$

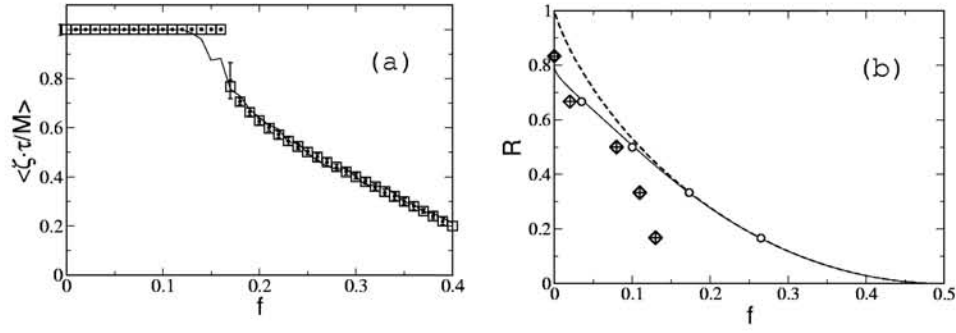

Figure 2: (a) Mean normalized overlap between the actual noise vector $\zeta$ and decoded noise $\widehat{\tau}$ for $K = 4$ and $C = 3$ (therefore $R = 1/4$). Theoretical values ($\square$), experimental averages over 20 runs for code word lengths $M = 5000$ ($\bullet$) and $M = 100$ (full line). (b) Transitions for $K = 6$. Shannon's bound (dashed line), information theory upper bound (full line) and thermodynamic transition obtained numerically ($\circ$). Theoretical ($\diamond$) and experimental ($+$, $M = 5000$ averaged over 20 runs) PP decoding transitions are also shown. In both figures, symbols are chosen larger than the error bars.

$$- \sum_{j=1}^{M} \ln \left[ e^F \prod_{\mu \in \mathcal{M}(j)} (1 + \widehat{m}_{\mu j}) + e^{-F} \prod_{\mu \in \mathcal{M}(j)} (1 - \widehat{m}_{\mu j}) \right]$$

The iteration of the maps (8) is actually one out of many different methods of finding extrema of this free-energy (not necessarily stable). This observation opens an alternative way for analyzing the performance of a decoding algorithm by studying the landscape (10).

## 4 Typical performance

### 4.1 Macroscopic description

The typical macroscopic states of the system during decoding can be described by histograms of the variables $m_{\mu k}$ and $\widehat{m}_{\mu k}$ averaged over all possible realizations of the noise vector $\zeta$. By applying the gauge transformation $\mathcal{J}_\mu \mapsto 1$ and $\tau_j \mapsto \tau_j \zeta_j$, assigning the probability distributions $P_0(x)$ to boundary fields and averaging over random local fields $F\zeta$ one obtains from (7) the recursion relation in the space of probability distributions $P(x)$:

$$P_n(x) = \int \prod_{l=1}^{C-1} d\widehat{x}_l \, \widehat{P}_{n-1}(\widehat{x}_l) \left\langle \delta \left[ x - F\zeta - \sum_{l=1}^{C-1} \widehat{x}_l \right] \right\rangle_\zeta$$

$$\widehat{P}_{n-1}(\widehat{x}) = \int \prod_{j=1}^{K-1} dx_j \, P_{n-1}(x_j) \, \delta \left[ \widehat{x} - \operatorname{atanh} \left( \prod_{j=1}^{K-1} \tanh(x_j) \right) \right], \quad (11)$$

where $P_n(x)$ is the distribution of effective fields at the $n$-th generation due to the previous generations and external fields, in the thermodynamic limit the distribution far from the boundary will be $P_\infty(x)$ (generation $n \to \infty$). The local field distribution at the central site is computed by replacing $C - 1$ by $C$ in (11), taking into account $C$ polygons in the generation just before the central site, and inserting the distribution $P_\infty(x)$. Equations (11) are identical to those obtained by the replica symmetric theory as in [12].

By setting initial (boundary) conditions $P_0(x)$ and numerically iterating (11), for $C \geq 3$ one can find, up to some noise level $f_s$, a single stable fixed point at infinite fields, corresponding to a totally aligned state (successful decoding). At $f_s$ a bifurcation occurs and two other fixed points appear, stable and unstable, the former corresponding to a misaligned state (decoding failure). This situation is identical to that one observed in [12]. In terms of the free-energy (10), below $f_s$ the landscape is dominated by the aligned state that is the global minimum. Above $f_s$ a sub-optimal state, corresponding to an exponentially large number of spurious local minima of the Hamiltonian (2), appears and convergence to the totally aligned state becomes a difficult task. At some critical noise level the totally aligned state loses the status of global minimum and the thermodynamic transition occurs.

The practical PP decoding is performed by setting initial conditions as $m_{\mu j} = 1 - 2f$, corresponding to the prior probabilities and iterating (8), until stationarity or a maximum number of iterations is attained. The estimate for the noise vector is then produced by computing $\hat{\tau}_j = \mathrm{sign}(m_j)$. At each decoding step the system can be described by histograms of the variables (8), this is equivalent to iterating (11) (a similar idea was presented in [7]). Below $f_s$ the process always converges to the successful decoding state, above $f_s$ it converges to the successful decoding only if the initial conditions are fine tuned, in general the process converges to the failure state. In Fig.2a we show the theoretical mean overlap between actual noise $\zeta$ and the estimate $\hat{\tau}$ as a function of the noise level $f$, as well as results obtained with PP decoding.

Information theory provides an upper bound for the maximum attainable code rate by equalizing the maximal information contents of the syndrome vector $z$ and of the noise estimate $\hat{\tau}$ [7]. The thermodynamic phase transition obtained by finding the stable fixed points of (11) and their free-energies interestingly coincides with this upper bound within the precision of the numerical calculation. Note that the performance predicted by thermodynamics is not practical as it requires $\mathcal{O}(2^M)$ operations for an exhaustive search for the global minimum of the free-energy. In Fig.2b we show the thermodynamic transition for $K = 6$ and compare with the upper bound, Shannon's bound and the theoretical $f_s$ values.

### 4.2 Tree-like approximation and the thermodynamic limit

The geometrical structure of a Gallager code defined by the matrix $A$ can be represented by a bipartite graph (*Tanner graph*) [16] with bit and check nodes. Each column $j$ of $A$ represents a bit node and each row $\mu$ represents a check node, $A_{\mu j} = 1$ means that there is an edge linking bit $j$ to check $\mu$. It is possible to show that for a random ensemble of regular codes, the probability of completing a cycle after walking $l$ edges starting from an arbitrary node is upper bounded by $\mathcal{P}[l; K, C, M] \leq l^2 K^l / M$. It implies that for very large $M$ only cycles of at least order $\ln M$ survive. In the thermodynamic limit $M \to \infty$ the probability $\mathcal{P}[l; K, C, M] \to 0$ for any finite $l$ and the bulk of the system is effectively tree-like. By mapping each check node to a polygon with $K$ bit nodes as vertices, one can map a Tanner graph into a Husimi lattice that is effectively a tree for any number of generations of order less than $\ln M$. It is experimentally observed that the number of iterations of (8) required for convergence does not scale with the system size, therefore, it is expected that the interior of a tree-like lattice approximates a Gallager code with increasing accuracy as the system size increases. Fig.2a shows that the approximation is fairly good even for sizes as small as $M = 100$.

## 5 Conclusions

To summarize, we solved exactly, without resorting to the replica method, a system representing a Gallager code on a Husimi cactus. The results obtained are in agreement with the replica symmetric calculation and with numerical experiments carried out in systems

of moderate size. The framework can be easily extended to MN and similar codes. New insights on the decoding process are obtained by looking at a proper free-energy landscape. We believe that methods of statistical physics are complimentary to those used in the statistical inference community and can enhance our understanding of general graphical models.

**Acknowledgments**

We acknowledge support from EPSRC (GR/N00562), The Royal Society (RV,DS) and from the JSPS RFTF program (YK).

**References**

[1] Plefka, T., (1982) Convergence condition of the TAP equation for the infinite-ranged Ising spin glass model. *Journal of Physics A* **15**, 1971-1978.

[2] Tanaka, T., Information geometry of mean field approximation. *to appear in Neural Computation*

[3] Saul, L.K. & , M.I. Jordan (1996) Exploiting tractable substructures in intractable. In Touretzky, D. S. , M. C. Mozer and M. E. Hasselmo (eds.), *Advances in Neural Information Processing Systems 8*, pp. 486-492. Cambridge, MA: MIT Press.

[4] Frey, B.J. & D.J.C. MacKay (1998) A revolution: belief propagation in graphs with cycles. In Jordan, M.I., M. J. Kearns and S.A. Solla (eds.), *Advances in Neural Information Processing Systems 10*, pp. 479-485 . Cambridge, MA: MIT Press.

[5] Berrou, C. & A. Glavieux (1996) Near optimum error correcting coding and decoding: Turbo-codes, *IEEE Transactions on Communications* **44**, 1261-1271.

[6] Gallager, R.G. (1963) Low-density parity-check codes, MIT Press, Cambridge, MA.

[7] MacKay, D.J.C. (1999) Good error-correcting codes based on very sparse matrices, *IEEE Transactions on Information Theory* **45**, 399-431.

[8] Kanter, I. & D. Saad (2000) Finite-size effects and error-free communication in Gaussian channels, *Journal of Physics A* **33**, 1675-1681.

[9] Sourlas, N. (1989) Spin-glass models as error-correcting codes, *Nature* **339**, 693-695.

[10] Derrida, B. (1981) Random-energy model: an exactly solvable model of disordered systems, *Physical Review B* **24**(5), 2613-2626.

[11] Vicente, R., D. Saad & Y. Kabashima (1999) Finite-connectivity systems as error-correcting codes, *Physical Review E* **60**(5), 5352-5366.

[12] Kabashima, Y., T. Murayama & D.Saad (2000) Typical performance of Gallager-type error-correcting codes, *Physical Review Letters* **84** (6), 1355-1358.

[13] Montanari, A. & N. Sourlas (2000) The statistical mechanics of turbo codes, *European Physical Journal B* **18**, 107-119.

[14] Sherrington, D. & K.Y.M. Wong (1987) Graph bipartitioning and the Bethe spin glass, *Journal of Physics A* **20**, L785-L791.

[15] Kabashima, Y. & D. Saad (1998) Belief propagation *vs.* TAP for decoding corrupted messages, *Europhysics Letters* **44** (5), 668-674.

[16] Kschischang, F.R. & B.J. Frey, (1998) Iterative decoding of compound codes by probability probagation in graphical models, *IEEE Journal on Selected Areas in Comm.* **16** (2), 153-159.

[17] Gujrati, P.D. (1995) Bethe or Bethe-like lattice calculations are more reliable than conventional mean-field calculations, *Physical Review Letters* **74** (5), 809-812.

[18] Rieger, H. & T.R. Kirkpatrick (1992) Disordered $p$-spin interaction models on Husimi trees, *Physical Review B* **45** (17), 9772-9777.
